# Non-Boltzmann Dynamics in Networks of Spiking Neurons

**Michael C. Crair and William Bialek**
Department of Physics, and
Department of Molecular and Cell Biology
University of California at Berkeley
Berkeley, CA 94720

## ABSTRACT

We study networks of spiking neurons in which spikes are fired as a Poisson process. The state of a cell is determined by the instantaneous firing rate, and in the limit of high firing rates our model reduces to that studied by Hopfield. We find that the inclusion of spiking results in several new features, such as a noise-induced asymmetry between "on" and "off" states of the cells and probability currents which destroy the usual description of network dynamics in terms of energy surfaces. Taking account of spikes also allows us to calibrate network parameters such as "synaptic weights" against experiments on real synapses. Realistic forms of the post synaptic response alters the network dynamics, which suggests a novel dynamical learning mechanism.

## 1  INTRODUCTION

In 1943 McCulloch and Pitts introduced the concept of two-state (binary) neurons as elementary building blocks for neural computation. They showed that essentially any finite calculation can be done using these simple devices. Two-state neurons are of questionable biological relevance, yet much of the subsequent work on modeling of neural networks has been based on McCulloch-Pitts type neurons because the two-state simplification makes analytic theories more tractable. Hopfield (1982, 1984)

showed that an asynchronous model of symmetrically connected two-state neurons was equivalent to Monte-Carlo dynamics on an 'energy' surface at zero temperature. The idea that the computational abilities of a neural network can be understood from the structure of an effective energy surface has been the central theme in much recent work.

In an effort to understand the effects of noise, Amit, Gutfreund and Sompolinsky (Amit *et al.*, 1985a; 1985b) assumed that Hopfield's 'energy' could be elevated to an energy in the statistical mechanics sense, and solved the Hopfield model at finite temperature. The problem is that the noise introduced in equilibrium statistical mechanics is of a very special form, and it is not clear that the stochastic properties of real neurons are captured by postulating a Boltzmann distribution on the energy surface.

Here we try to do a slightly more realistic calculation, describing interactions among neurons through action potentials which are fired according to probabilistic rules. We view such calculations as intermediate between the purely phenomenological treatment of neural noise by Amit *et al.* and a fully microscopic description of neural dynamics in terms of ion channels and their associated noise. We find that even our limited attempt at biological realism results in some interesting deviations from previous ideas on network dynamics.

## 2   THE MODEL

We consider a model where neurons have a continuous firing rate, but the generation of action potentials is a Poisson process. This means that the "state" of each cell $i$ is described by the instantaneous rate $r_i(t)$, and the probability that this cell will fire in a time interval $[t, t + dt]$ is given by $r_i(t)dt$. Evidence for the near-Poisson character of neuronal firing can be found in the mammalian auditory nerve (Siebert, 1965; 1968), and retinal ganglion cells (Teich *et al.*, 1978, Teich and Saleh, 1981). To stay as close as possible to existing models, we assume that the rate $r(t)$ of a neuron is a sigmoid function, $g(x) = 1/(1+e^{-x})$, of the total input $x$ to the neuron.

The input is assumed to be a weighted sum of the spikes received from all other neurons, so that

$$r_i(t) = r_m g \left[ \sum_{\mu} \sum_{j} J_{ij} f(t - t_j^{\mu}) - \Theta_i \right] . \tag{1}$$

$J_{ij}$ is the matrix of connection strengths between neurons, $r_m$ is the maximum spike rate of the neuron, and $\Theta_i$ is the neuronal threshold. $f(t)$ is a time weighting function, corresponding schematically to the time course of post-synaptic currents injected by a pre-synaptic spike; a good first order approximation for this function is $f(t) \sim e^{-t/\tau}$, but we also consider functions with more than one time constant. (Aidley, 1980, Fetz and Gustafsson, 1983).

We can think of the spike train from the $i^{th}$ neuron, $\sum_{\mu} \delta(t - t_i^{\mu})$, as an approximation to the true firing rate $r_i(t)$; of course this approximation improves as the

spikes come closer together at high firing rates. If we write

$$\sum_{\mu} \delta(t - t_i^{\mu}) = r_i(t) + \eta_i(t) \tag{2}$$

we have defined the noise $\eta_i$ in the spike train. The equations of motion for the rates then become

$$r_i(t) = r_m g \left[ \sum_j J_{ij} f \circ r_j(t) - \Theta_i + N_i(t) \right], \tag{3}$$

where $N_i(t) = \sum_j J_{ij} \eta_j(t)$ and $f \circ r_j(t)$ is the convolution of $f(t)$ with the spike rate $r_j(t)$. The statistics of the fluctuations in the spike rate $\eta_j(t)$ are $\langle \eta_j(t) \rangle = 0$, $\langle \eta_i(t)\eta_j(t') \rangle = \delta_{ij}(t - t')r_j(t)$.

## 3   DYNAMICS

If the post-synaptic response $f(t)$ is exactly exponential, we can invert Eq. (3) to obtain a first order equation for the normalized spike rate $y_i(t) \equiv r_i(t)/r_m$. More precise descriptions of the post-synaptic response will yield higher order time derivatives with coefficients that depend on the relative time constants in $f(t)$. We will comment later on the relevance of these higher order terms, but consider first the lowest order description. By inverting Eq. (3) we obtain a stochastic differential equation analogous to the Langevin equation describing Brownian motion:

$$\frac{dg^{-1}(y_i)}{dt} = -\frac{dE}{dy_i} + N_i(t), \tag{4a}$$

where the deterministic forces are given by

$$\frac{dE}{dy_i} = \frac{g^{-1}(y_i)}{\tau} - r_m \sum_j J_{ij}(y_j - 1/2). \tag{4b}$$

Note that Eq. (4) is nearly equivalent to the "charging equation" Hopfield (1984) assumed in his discussion of continuous neurons, except we have explicitly included the noise from the spikes. This system is precisely equivalent to the Hopfield two-state model in the limit of large spike rate ($r_m\tau \Rightarrow \infty, J_{ij} = $ constant), and no noise. In a thermodynamic system near equilibrium, the noise "force" $N_i(t)$ is related to the friction coefficient via the fluctuation dissipation theorem. In this system however, there is no analogous relationship.

A standard transformation, analogous to deriving Einstein's diffusion equation from the Langevin equation (Stratonovich, 1963, 1967), yields a probabilistic description for the evolution of the neural system, a form of Fokker-Planck equation for the time evolution of $P(\{y_i\})$, the probability that the network is in a state described by the normalized rates $\{y_i\}$; we write the Fokker-Planck equation below for a simple case.

A useful interpretation to consider is that the system, starting in a non-equilibrium state, diffuses or evolves in phase space, to a final stationary state.

We can make our description of the post-synaptic response $f(t)$ more accurate by including two (or more) exponential time constants, corresponding roughly to the rise and fall time of the post synaptic potential. This inclusion necessitates the addition of a second order term in the Langevin equation (Eq. 4). This is analogous to including an inertial term in a diffusive description, so that the system is no longer purely dissipative. This additional complication has some interesting consequences. Adjusting the relative length of the rise to fall time of the post synaptic potential effects the rate of relaxation to local equilibrium of the system. In order to perform most efficaciously as an associative memory, a neural system will "choose" critical damping time constants, so that relaxation is fastest. Thus, by adjusting the time course of the post synaptic potential, the system can "learn" of a local stationary state, without adjusting the synaptic strengths. This novel learning mechanism could be a form of fine tuning of already established memories, or could be a unique form of dynamical short-term memory.

## 4   QUALITATIVE RESULTS

In order to understand the dynamics of our Fokker-Planck equation, we begin by considering the case of two neurons interacting with each other. There are two limiting behaviors. If the neurons are weakly coupled ($J < J_c, J_c = 4/r_m\tau$), then the only stable state of the system is with both neurons firing at a mean firing rate, $\frac{1}{2}r_m$. If the neurons are strongly (and positively) coupled ($J > J_c$), then isolated basins of attraction, or stationary states are formed, one stationary state corresponding to both neurons being active, the other state has both neurons relatively (but not absolutely) quiescent. In the strong coupling limit, one can reduce the problem to motion along the a collective coordinate connecting the two stable states. The resulting one dimensional Fokker-Planck equation is

$$\frac{\partial}{\partial t}P(y,t) = \frac{\partial}{\partial y}\left[U'(y)P(y,t) + \frac{\partial}{\partial y}T(y)P(y,t)\right], \qquad (5)$$

where $U(y)$ is an effective potential energy,

$$U'(y) = y(1-y)[\frac{g^{-1}(y)}{\tau} - \frac{1}{2}r_m J(y - \frac{1}{2}) + \frac{1}{4}J^2 r_m y(3 - 5y)], \qquad (6)$$

and $T(y)$ is a spatially varying effective temperature, $T(y) = \frac{1}{8}J^2 r_m y^3(1-y)^2$. One can solve to find the size of the stable regions, and the stationary probability distribution,

$$P^s(y) = \frac{B}{T(y)}\exp\left[-\int_{\frac{1}{2}}^{y}\frac{U'(y)}{T(y)}dy\right]. \qquad (7)$$

We have done numerical simulations which confirm the qualitative predictions of the one dimensional Fokker-Planck equation. This analysis shows that the non-uniform

and asymmetric temperature distribution alters the relative stability of the stable states, in the favor of the 'off' state. This effect does have some biological pertinence, as it is well known that on average neurons are more likely to be quiescent then active. In our model the asymmetry is a direct consequence of the Poisson nature of the neuronal firing.

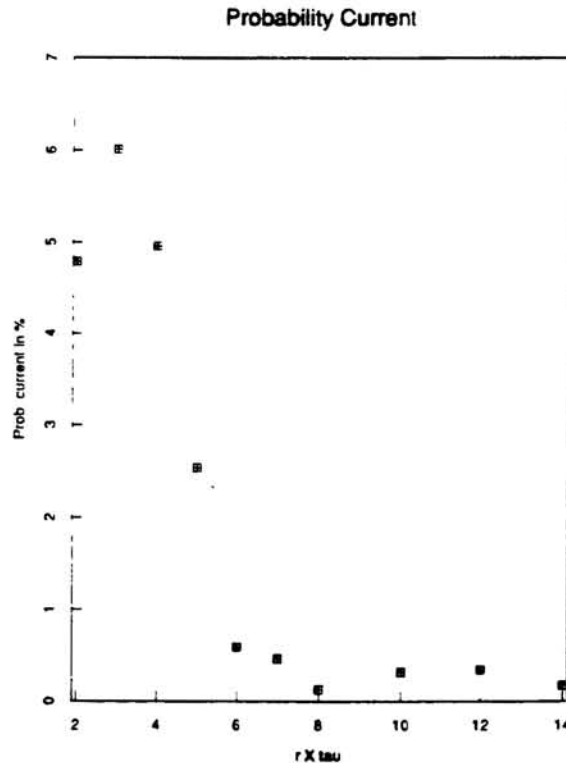

**Figure 1:** Probability current in the stationary state for two neurons that are strongly interacting. Computed as a ratio of the number of excess excursions in one direction to the total number of excursions, in percent. In thermodynamic equillibrium, detailed balance would force the current to be zero. Shown as a function of the number of spikes in an e-folding time of the post-synaptic response.

There are further surprises to be found in the simple two neuron model. Since the interaction between the neurons is not time reversal invariant, detailed balance is not maintained in the system. Thus, even the stationary probability distribution has non-zero probability current, so that the system tends to cycle probabilistically through state space. The presence of the current further alters the relative probability of the two stable states, as confirmed by numerical simulations, and renders the application of equilibrium statistical mechanics inappropriate.

Simulations also confirm (Fig. 1) that the probability current falls off with increasing maximum spike rate ($r_m \tau$), because the effective noise is suppressed when the spike rate is high. However, at biologically reasonable spike rates ($r_m \sim 150\,\mathrm{s}^{-1}$), the probability current is significant. These currents destroy any sense of a global

energy function or thermodynamic temperature.

One advantage of treating spikes explicitly is that we can relate the abstract synaptic strength $J$ to observable parameters. In Fig. 2 we compare $J$ with the experimentally accessible spike number to spike number transfer across the synapse, for a two neuron system. Note that critical coupling (see above) corresponds to a rather large value of $\sim 4/5^{th}$ of a spike emitted per spike received.

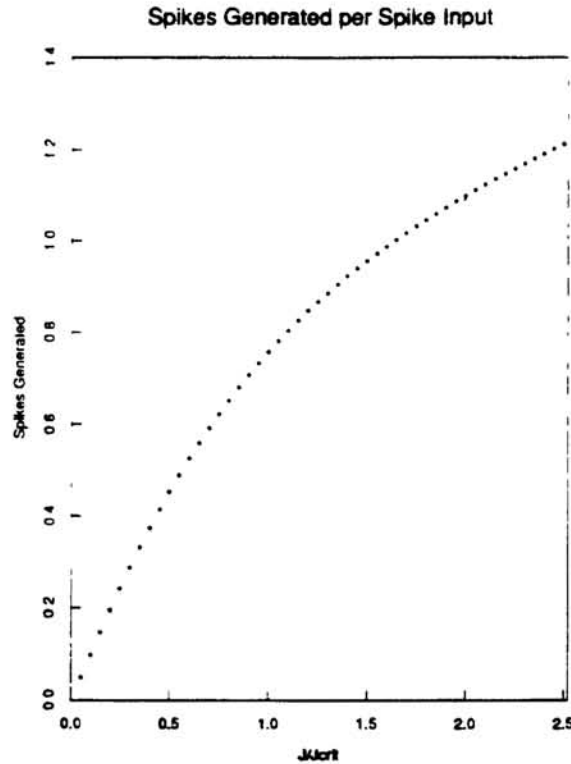

**Figure 2:** Single neuron spike response to the receipt of a spike from a coupled neuron. Since response is probabilistic, fractional spikes are relevant. Computed as a function of $J/J_{critical}$, where $J_{critical}$ is the minimum synaptic strength necessary for isolated basins of attraction.

Many of the simple ideas we have introduced for the two neuron system carry over to the multi-neuron case. If the matrix of connection strengths obeys the "Hebb" rule (often used to model associative memory),

$$J_{ij} = \frac{1}{N} J \sum_{\mu} \xi_i^{\mu} \xi_j^{\mu}, \qquad (8)$$

then a stability analysis yields the same critical value for the connection strength $J$ (note that we have scaled by $N$, and the sum on $\mu$ runs from 1 to $p$, the number of memories to be stored). Calculation of the spike-out/spike-in ratio for the multi-neuron system at critical coupling shows that it scales like $(\alpha/N)^{\frac{1}{2}}$, where $p = \alpha N$.

Since most neural systems naturally have a small spike-out/spike-in ratio, this (together with Fig. 2) suggests that small networks will have to be strongly driven in order to achieve isolated basins of attraction for "memories;" this is in agreement with the one available experiment (Kleinfeld *et al.*, 1990). In contrast, large networks achieve criticality with more natural spike to spike ratios. For instance, if a network of $10^4 - 10^5$ connected neurons is to have multiple stable "memory" states as in the original Hopfield model, we predict that a neuron needs to receive 100-500 contiguous action potentials to stimulate the emission of its own spike. This prediction agrees with experiments done on the hippocampus (McNaughton *et al.*, 1981), where about 400 convergent inputs are needed to discharge a granule cell.

# 5   CONCLUSIONS

To conclude, we will just summarize our major points:

- Spike noise generated by the Poisson firing of neurons breaks the symmetry between on/off states, in favor of the "off" state.

- State dependent spike noise also destroys any sense of a global energy function, let alone a thermodynamic 'temperature'. This makes us suspicious of attempts to apply standard techniques of statistical mechanics.

- By explicitly modeling the interaction of neurons via spikes, we have direct access to experiments which can guide, and be guided by our theory. Specifically, our theory predicts that for a given connection strength between neurons, larger networks of neurons will function as memories at naturally small spike-input to spike-output ratios.

- More realistic forms of post synaptic response to the receipt of action potentials alters the network dynamics. By adjusting the relative rise and fall time of the post-synaptic potential, the network speeds the relaxation to the local stable state. This implies that more efficacious memories, or "learning", can result without altering the strength of the synaptic weights.

Finally, we comment on the dynamics of networks in the $N \to \infty$ limit. We might imagine that some of the complexities we find in the two-neuron case would go away, in particular the probability currents. We have been able to prove that this does not happen in any rigorous sense for realistic forms of spike noise, although in practice the currents may become small. The function of the network as a memory (for example) would then depend on a clean separation of time scales between relaxation into a single basin of attraction and noise-driven transitions to neighboring basins. Arranging for this separation of time scales requires some constraints on synaptic connectivity and firing rates which might be testable in experiments on real circuits.

## References

D. J. Aidley (1980), *Physiology of Excitable Cells, 2nd Edition*, Cambridge University Press, Cambridge.

D. J. Amit, H. Gutfreund and H. Sompolinsky (1985a), *Phys. Rev. A*, 2, 1007-1018.

D. J. Amit, H. Gutfreund and H. Sompolinsky (1985b), *Phys. Rev. Lett.*, 55, 1530-1533.

E. E. Fetz and B. Gustafsson (1983), *J. Physiol.*, **341**, 387.

J. J. Hopfield (1982), *Proc. Nat. Acad. Sci. USA*, 79, 2554-2558.

J. J. Hopfield (1984), *Proc. Nat. Acad. Sci. USA*, 81, 3088-3092.

D. Kleinfeld, F. Raccuia-Behling, and H. J. Chiel (1990), *Biophysical Journal*, in press.

W. S. McCulloch and W. Pitts (1943), *Bull. of Math. Biophys.*, 5, 115-133.

B. L. McNaughton, C. A. Barnes and P. Anderson (1981), *J. Neurophysiol.* 46, 952-966.

W. M. Siebert (1965), *Kybernetik*, 2, 206.

W. M. Siebert (1968) in *Recognizing Patterns*, p104, P.A. Kohlers and M. Eden, Eds., MIT Press, Cambridge.

R. L. Stratonovich (1963,1967), *Topics in the Theory of Random Noise*, Vol. I and II, Gordon & Breach, New York.

M. C. Teich, L. Martin and B.I. Cantor (1978), *J. Opt. Soc. Am.*, 68, 386.

M. C. Teich and B.E.A. Saleh (1981), *J. Opt. Soc. Am.*,71, 771.